# Convergence of Indirect Adaptive Asynchronous Value Iteration Algorithms

**Vijaykumar Gullapalli**
Department of Computer Science
University of Massachusetts
Amherst, MA 01003
vijay@cs.umass.edu

**Andrew G. Barto**
Department of Computer Science
University of Massachusetts
Amherst, MA 01003
barto@cs.umass.edu

## Abstract

Reinforcement Learning methods based on approximating dynamic programming (DP) are receiving increased attention due to their utility in forming reactive control policies for systems embedded in dynamic environments. Environments are usually modeled as controlled Markov processes, but when the environment model is not known a priori, adaptive methods are necessary. Adaptive control methods are often classified as being direct or indirect. Direct methods directly adapt the control policy from experience, whereas indirect methods adapt a model of the controlled process and compute control policies based on the latest model. Our focus is on indirect adaptive DP-based methods in this paper. We present a convergence result for indirect adaptive asynchronous value iteration algorithms for the case in which a look-up table is used to store the value function. Our result implies convergence of several existing reinforcement learning algorithms such as adaptive real-time dynamic programming (ARTDP) (Barto, Bradtke, & Singh, 1993) and prioritized sweeping (Moore & Atkeson, 1993). Although the emphasis of researchers studying DP-based reinforcement learning has been on direct adaptive methods such as Q-Learning (Watkins, 1989) and methods using TD algorithms (Sutton, 1988), it is not clear that these direct methods are preferable in practice to indirect methods such as those analyzed in this paper.

# 1   INTRODUCTION

Reinforcement learning methods based on approximating dynamic programming (DP) are receiving increased attention due to their utility in forming reactive control policies for systems embedded in dynamic environments. In most of this work, learning tasks are formulated as Markovian Decision Problems (MDPs) in which the environment is modeled as a controlled Markov process. For each observed environmental state, the agent consults a policy to select an action, which when executed causes a probabilistic transition to a successor state. State transitions generate rewards, and the agent's goal is to form a policy that maximizes the expected value of a measure of the long-term reward for operating in the environment. (Equivalent formulations minimize a measure of the long-term cost of operating in the environment.) Artificial neural networks are often used to store value functions produced by these algorithms (e.g., (Tesauro, 1992)).

Recent advances in reinforcement learning theory have shown that asynchronous value iteration provides an important link between reinforcement learning algorithms and classical DP methods for value iteration (VI) (Barto, Bradtke, & Singh, 1993). Whereas conventional VI algorithms use repeated exhaustive "sweeps" of the MDP's state set to update the value function, asynchronous VI can achieve the same result without proceeding in systematic sweeps (Bertsekas & Tsitsiklis, 1989). If the state ordering of an asynchronous VI computation is determined by state sequences generated during real or simulated interaction of a controller with the Markov process, the result is an algorithm called *Real-Time DP* (RTDP) (Barto, Bradtke, & Singh, 1993). Its convergence to optimal value functions in several kinds of problems follows from the convergence properties of asynchronous VI (Barto, Bradtke, & Singh, 1993).

# 2   MDPS WITH INCOMPLETE INFORMATION

Because asynchronous VI employs a basic update operation that involves computing the expected value of the next state for all possible actions, it requires a complete and accurate model of the MDP in the form of state-transition probabilities and expected transition rewards. This is also true for the use of asynchronous VI in RTDP. Therefore, when state-transition probabilities and expected transition rewards are not completely known, asynchronous VI is not directly applicable. Problems such as these, which are called MDPs with incomplete information,[1] require more complex *adaptive* algorithms for their solution. An *indirect* adaptive method works by identifying the underlying MDP via estimates of state transition probabilities and expected transition rewards, whereas a *direct* adaptive method (e.g., Q-Learning (Watkins, 1989)) adapts the policy or the value function without forming an explicit model of the MDP through system identification.

In this paper, we prove a convergence theorem for a set of algorithms we call *indirect adaptive asynchronous VI* algorithms. These are indirect adaptive algorithms that result from simply substituting current estimates of transition probabilities and expected transition rewards (produced by some concurrently executing identification

algorithm) for their actual values in the asynchronous value iteration computation. We show that under certain conditions, indirect adaptive asynchronous VI algorithms converge with probability one to the optimal value function. Moreover, we use our result to infer convergence of two existing DP-based reinforcement learning algorithms, adaptive real-time dynamic programming (ARTDP) (Barto, Bradtke, & Singh, 1993), and prioritized sweeping (Moore & Atkeson, 1993).

## 3   CONVERGENCE OF INDIRECT ADAPTIVE ASYNCHRONOUS VI

Indirect adaptive asynchronous VI algorithms are produced from non-adaptive algorithms by substituting a current approximate model of the MDP for the true model in the asynchronous value iteration computations. An indirect adaptive algorithm can be expected to converge only if the corresponding non-adaptive algorithm, with the true model used in the place of each approximate model, converges. We therefore restrict attention to indirect adaptive asynchronous VI algorithms that correspond in this way to convergent non-adaptive algorithms. We prove the following theorem:

**Theorem 1** *For any finite state, finite action MDP with an infinite-horizon discounted performance measure, any indirect adaptive asynchronous VI algorithm (for which the corresponding non-adaptive algorithm converges) converges to the optimal value function with probability one if*
*1) the conditions for convergence of the non-adaptive algorithm are met,*
*2) in the limit, every action is executed from every state infinitely often, and*
*3) the estimates of the state-transition probabilities and the expected transition rewards remain bounded and converge in the limit to their true values with probability one.*

**Proof** The proof is given in Appendix A.2.

## 4   DISCUSSION

Condition 2 of the theorem, which is also required by direct adaptive methods to ensure convergence, is usually unavoidable. It is typically ensured by using a stochastic policy. For example, we can use the Gibbs distribution method for selecting actions used by Watkins (1989) and others. Given condition 2, condition 3 is easily satisfied by most identification methods. In particular, the simple maximum-likelihood identification method (see Appendix A.1, items 6 and 7) converges to the true model with probability one under this condition.

Our result is valid only for the special case in which the value function is explicitly stored in a look-up table. The case in which general function approximators such as neural networks are used requires further analysis.

Finally, an important issue not addressed in this paper is the trade-off between system identification and control. To ensure convergence of the model, all actions have to be executed infinitely often in every state. On the other hand, on-line control objectives are best served by executing the action in each state that is optimal according to the current value function (i.e., by using the certainty equivalence

optimal policy). This issue has received considerable attention from control theorists (see, for example, (Kumar, 1985), and the references therein). Although we do not address this issue in this paper, for a specific estimation method, it may be possible to determine an action selection scheme that makes the best trade-off between identification and control.

# 5    EXAMPLES OF INDIRECT ADAPTIVE ASYNCHRONOUS VI

One example of an indirect adaptive asynchronous VI algorithm is ARTDP (Barto, Bradtke, & Singh, 1993) with maximum-likelihood identification. In this algorithm, a randomized policy is used to ensure that every action has a non-zero probability of being executed in each state. The following theorem for ARDTP follows directly from our result and the corresponding theorem for RTDP in (Barto, Bradtke, & Singh, 1993):

**Theorem 2** *For any discounted MDP and any initial value function, trial-based[2] ARTDP converges with probability one.*

As a special case of the above theorem, we can obtain the result that in similar problems the prioritized sweeping algorithm of Moore and Atkeson (Moore & Atkeson, 1993) converges to the optimal value function. This is because prioritized sweeping is a special case of ARTDP in which states are selected for value updates based on their priority and the processing time available. A state's priority reflects the utility of performing an update for that state, and hence prioritized sweeping can improve the efficiency of asynchronous VI. A similar algorithm, Queue-Dyna (Peng & Williams, 1992), can also be shown to converge to the optimal value function using a simple extension of our result.

# 6    CONCLUSIONS

We have shown convergence of indirect adaptive asynchronous value iteration under fairly general conditions. This result implies the convergence of several existing DP-based reinforcement learning algorithms. Moreover, we have discussed possible extensions to our result. Our result is a step toward a better understanding of indirect adaptive DP-based reinforcement learning methods. There are several promising directions for future work.

One is to analyze the trade-off between model estimation and control mentioned earlier to determine optimal methods for action selection and to integrate our work with existing results on adaptive methods for MDPs (Kumar, 1985). Second, analysis is needed for the case in which a function approximation method, such as a neural network, is used instead of a look-up table to store the value function. A third possible direction is to analyze indirect adaptive versions of more general DP-based algorithms that combine asynchronous policy iteration with asynchronous

policy evaluation. Several non-adaptive algorithms of this nature have been proposed recently (e.g., (Williams & Baird, 1993; Singh & Gullapalli)).

Finally, it will be useful to examine the relative efficacies of direct and indirect adaptive methods for solving MDPs with incomplete information. Although the emphasis of researchers studying DP-based reinforcement learning has been on direct adaptive methods such as Q-Learning and methods using TD algorithms, it is not clear that these direct methods are preferable in practice to indirect methods such as the ones discussed here. For example, Moore and Atkeson (1993) report several experiments in which prioritized sweeping significantly outperforms Q-learning in terms of the computation time and the number of observations required for convergence. More research is needed to characterize circumstances for which the various reinforcement learning methods are best suited.

# APPENDIX

## A.1 NOTATION

1. Time steps are denoted $t = 1, 2, \ldots$, and $x_t$ denotes the last state observed before time $t$. $x_t$ belongs to a finite state set $S = \{1, 2, \ldots, n\}$.

2. Actions in a state are selected according to a policy $\pi$, where $\pi(i) \in A$, a finite set of actions, for $1 \leq i \leq n$.

3. The probability of making a transition from state $i$ to state $j$ on executing action $a$ is $p^a(i, j)$.

4. The expected reward from executing action $a$ in state $i$ is $r(i, a)$. The reward received at time $t$ is denoted $r_t(x_t, a_t)$.

5. $0 \leq \gamma < 1$ is the discount factor.

6. Let $\hat{p}_t^a(i, j)$ denote the estimate at time $t$ of the probability of transition from state $i$ to $j$ on executing action $a \in A$. Several different methods can be used for estimating $\hat{p}_t^a(i, j)$. For example, if $n_t^a(i, j)$ is the observed number of times before time step $t$ that execution of action $a$ when the system was in state $i$ was followed by a transition to state $j$, and $n_t^a(i) = \sum_{j \in S} n_t^a(i, j)$ is the number of times action $a$ was executed in state $i$ before time step $t$, then, for $1 \leq i \leq n$ and for all $a \in A$, the maximum-likelihood state-transition probability estimates at time $t$ are

$$\hat{p}_t^a(i, j) = \frac{n_t^a(i, j)}{n_t^a(i)}, \quad 1 \leq j \leq n.$$

Note that the maximum-likelihood estimates converge to their true values with probability one if $n_t^a(i) \rightarrow \infty$ as $t \rightarrow \infty$, i.e., every action is executed from every state infinitely often.

Let $p^a(i) = [p^a(i, 1), \ldots, p^a(i, n)] \in [0, 1]^n$, and similarly, $\hat{p}_t^a(i) = [\hat{p}_t^a(i, 1), \ldots, \hat{p}_t^a(i, n)] \in [0, 1]^n$. We will denote the $|S| \times |A|$ matrix of transition probabilities associated with state $i$ by $P(i)$ and its estimate at time $t$ by $\hat{P}_t(i)$. Finally, $P$ denotes the vector of matrices $[P(1), \ldots, P(n)]$, and $\hat{P}_t$ denotes the vector $[\hat{P}_t(1), \ldots, \hat{P}_t(n)]$.

7. Let $\hat{r}_t(i, a)$ denote the estimate at time $t$ of the expected reward $r(i, a)$, and let $\hat{r}_t$ denote all the $|S| \times |A|$ estimates at time $t$. Again, if maximum-likelihood estimation is used,

$$\hat{r}_t(i, a) = \frac{\sum_{k=1}^{t} r_k(x_k, a_k) I_{ia}(x_k, a_k)}{n_t^a(i)},$$

where $I_{ia} : S \times A \rightarrow \{0, 1\}$ is the indicator function for the state-action pair $i, a$.

8. $V_t^*$ denotes the optimal value function for the MDP defined by the estimates $\hat{P}_t$ and $\hat{r}_t$ of $P$ and $r$ at time $t$. Thus, $\forall i \in S$,

$$V_t^*(i) = \max_{a \in A}\{\hat{r}_t(i, a) + \gamma \sum_{j \in S} \hat{p}_t^a(i, j) V_t^*(j)\}.$$

Similarly, $V^*$ denotes the optimal value function for the MDP defined by $P$ and $r$.

9. $B_t \subseteq S$ is the subset of states whose values are updated at time $t$. Usually, at least $x_t \in B_t$.

## A.2 PROOF OF THEOREM 1

In indirect adaptive asynchronous VI algorithms, the estimates of the MDP parameters at time step $t$, $\hat{P}_t$ and $\hat{r}_t$, are used in place of the true parameters, $P$ and $r$, in the asynchronous VI computations at time $t$. Hence the value function is updated at time $t$ as

$$V_{t+1}(i) = \begin{cases} \max_{a \in A}\{\hat{r}_t(i, a) + \gamma \sum_{j \in S} \hat{p}_t^a(i, j) V_t(j)\} & \text{if } i \in B_t \\ V_t(i) & \text{otherwise,} \end{cases}$$

where $B(t) \subseteq S$ is the subset of states whose values are updated at time $t$.

First note that because $\hat{P}_t$ and $\hat{r}_t$ are assumed to be bounded for all $t$, $V_t$ is also bounded for all $t$. Next, because the optimal value function given the model $\hat{P}_t$ and $\hat{r}_t$, $V_t^*$, is a continuous function of the estimates $\hat{P}_t$ and $\hat{r}_t$, convergence of these estimates w.p. 1 to their true values implies that

$$V_t^* \xrightarrow{w.p.\ 1} V^*,$$

where $V^*$ is the optimal value function for the original MDP. The convergence w.p. 1 of $V_t^*$ to $V^*$ implies that given an $\epsilon > 0$ there exists an integer $T > 0$ such that for all $t \geq T$,

$$\|V_t^* - V^*\| < \frac{(1 - \gamma)}{2\gamma} \epsilon \quad \text{w.p. 1.} \tag{1}$$

Here, $\| . \|$ can be any norm on $\Re^n$, although we will use the $\ell^\infty$ or max norm.

In algorithms based on asynchronous VI, the values of only the states in $B_t \subseteq S$ are updated at time $t$, although the value of each state is updated infinitely often. For an arbitrary $x \in S$, let us define the infinite subsequence $\{t_k^x\}_{k=0}^\infty$ to be the times when the value of state $x$ gets updated. Further, let us only consider updates at, or after, time $T$, where $T$ is from equation (1) above, so that $t_0^x \geq T$ for all $x \in S$.

By the nature of the VI computation we have, for each $t \geq 1$,

$$|V_{t+1}(i) - V_t^*(i)| \leq \gamma \|V_t - V_t^*\| \quad \text{if } i \in B_t. \tag{2}$$

Using inequality (2), we can get a bound for $|V_{t_k^\infty+1}(x) - V_{t_k^\infty}^*(x)|$ as

$$|V_{t_k^\infty+1}(x) - V_{t_k^\infty}^*(x)| < \gamma^{k+1} \|V_{t_0^\infty} - V_{t_0^\infty}^*\| + (1 - \gamma^k)\epsilon \quad \text{w.p. 1.} \tag{3}$$

We can verify that the bound in (3) is correct through induction. The bound is clearly valid for $k = 0$. Assuming it is valid for $k$, we show that it is valid for $k + 1$:

$$
\begin{aligned}
|V_{t_{k+1}^\infty+1}(x) - V_{t_{k+1}^\infty}^*(x)| &\leq \gamma \|V_{t_{k+1}^\infty} - V_{t_{k+1}^\infty}^*\| \\
&\leq \gamma(\|V_{t_{k+1}^\infty} - V_{t_k^\infty}^*\| + \|V_{t_k^\infty}^* - V_{t_{k+1}^\infty}^*\|) \\
&< \gamma |V_{t_{k+1}^\infty}(x) - V_{t_k^\infty}^*(x)| + \gamma\left(\frac{(1-\gamma)}{\gamma}\epsilon\right) \quad \text{w.p. 1} \\
&= \gamma |V_{t_k^\infty+1}(x) - V_{t_k^\infty}^*(x)| + (1-\gamma)\epsilon \\
&< \gamma(\gamma^{k+1} \|V_{t_0^\infty} - V_{t_0^\infty}^*\| + (1-\gamma^k)\epsilon) + (1-\gamma)\epsilon \quad \text{w.p. 1} \\
&= \gamma^{k+2} \|V_{t_0^\infty} - V_{t_0^\infty}^*\| + (1-\gamma^{k+1})\epsilon.
\end{aligned}
$$

Taking the limit as $k \to \infty$ in equation (3) and observing that for each $x$, $\lim_{k\to\infty} V_{t_k^\infty}^*(x) = V^*(x)$ w.p. 1, we obtain

$$\lim_{k\to\infty} |V_{t_k^\infty+1}(x) - V^*(x)| < \epsilon \quad \text{w.p. 1.}$$

Since $\epsilon$ and $x$ are arbitrary, this implies that $V_t \to V^*$ w.p. 1.    $\square$

## Acknowledgements

We gratefully acknowledge the significant contribution of Peter Dayan, who pointed out that a restrictive condition for convergence in an earlier version of our result was actually unnecessary. This work has also benefited from several discussions with Satinder Singh. We would also like to thank Chuck Anderson for his timely help in preparing this material for presentation at the conference. This material is based upon work supported by funding provided to A. Barto by the AFOSR, Bolling AFB, under Grant AFOSR-F49620-93-1-0269 and by the NSF under Grant ECS-92-14866.

## Footnotes

[1]These problems should not be confused with MDPs with incomplete *state* information, i.e., partially observable MDPs.

[2]As in (Barto, Bradtke, & Singh, 1993), by *trial-based* execution of an algorithm we mean its use in an infinite series of trials such that every state is selected infinitely often to be the start state of a trial.

## References

[1] A.G. Barto, S.J. Bradtke, and S.P. Singh. Learning to act using real-time dynamic programming. Technical Report 93-02, University of Massachusetts, Amherst, MA, 1993.

[2] D. P. Bertsekas and J. N. Tsitsiklis. *Parallel and Distributed Computation: Numerical Methods.* Prentice-Hall, Englewood Cliffs, NJ, 1989.

[3] P. R. Kumar. A survey of some results in stochastic adaptive control. *SIAM Journal of Control and Optimization*, 23(3):329–380, May 1985.

[4] A. W. Moore and C. G. Atkeson. Memory-based reinforcement learning: Efficient computation with prioritized sweeping. In S. J. Hanson, J. D. Cowan, and C. L. Giles, editors, *Advances in Neural Information Processing Systems 5*, pages 263–270, San Mateo, CA, 1993. Morgan Kaufmann Publishers.

[5] J. Peng and R. J. Williams. Efficient learning and planning within the dyna framework. In *Proceedings of the Second International Conference on Simulation of Adaptive Behavior*, Honolulu, HI, 1992.

[6] S. P. Singh and V. Gullapalli. Asynchronous modified policy iteration with single-sided updates. (Under review).

[7] R. S. Sutton. Learning to predict by the methods of temporal differences. *Machine Learning*, 3:9–44, 1988.

[8] G. J. Tesauro. Practical issues in temporal difference learning. *Machine Learning*, 8(3/4):257–277, May 1992.

[9] C. J. C. H. Watkins. *Learning from delayed rewards*. PhD thesis, Cambridge University, Cambridge, England, 1989.

[10] R. J. Williams and L. C. Baird. Analysis of some incremental variants of policy iteration: First steps toward understanding actor-critic learning systems. Technical Report NU-CCS-93-11, Northeastern University, College of Computer Science, Boston, MA 02115, September 1993.